# Generalization Error Bounds for Aggregation by Mirror Descent with Averaging

**Anatoli Juditsky**
Laboratoire de Modélisation et Calcul - Université Grenoble I
B.P. 53, 38041 Grenoble, France
anatoli.iouditski@imag.fr

**Alexander Nazin**
Institute of Control Sciences - Russian Academy of Science
65, Profsoyuznaya str., GSP-7, Moscow, 117997, Russia
nazine@ipu.rssi.ru

**Alexandre Tsybakov**
Laboratoire de Probabilités et Modèles Aléatoires - Université Paris VI
4, place Jussieu, 75252 Paris Cedex, France
tsybakov@ccr.jussieu.fr

**Nicolas Vayatis**
Laboratoire de Probabilités et Modèles Aléatoires - Université Paris VI
4, place Jussieu, 75252 Paris Cedex, France
vayatis@ccr.jussieu.fr

## Abstract

We consider the problem of constructing an aggregated estimator from a finite class of base functions which approximately minimizes a convex risk functional under the $\ell_1$ constraint. For this purpose, we propose a stochastic procedure, the mirror descent, which performs gradient descent in the dual space. The generated estimates are additionally averaged in a recursive fashion with specific weights. Mirror descent algorithms have been developed in different contexts and they are known to be particularly efficient in high dimensional problems. Moreover their implementation is adapted to the online setting. The main result of the paper is the upper bound on the convergence rate for the generalization error.

## 1 Introduction

We consider the aggregation problem (cf. [16]) where we have at hand a *finite* class of $M$ predictors which are to be combined linearly under an $\ell_1$ constraint $\|\theta\|_1 = \lambda$ on the vector $\theta \in \mathbb{R}^M$ that determines the coefficients of the linear combination. In order to exhibit such a combination, we focus on the strategy of penalized convex risk minimization which

is motivated by recent statistical studies of boosting and SVM algorithms [11, 14, 18]. Moreover, we take a stochastic approximation approach which is particularly relevant in the online setting since it leads to recursive algorithms where the update uses a single data observation per iteration step. In this paper, we consider a general setting for which we propose a novel stochastic gradient algorithm and show tight upper bounds on its expected accuracy. Our algorithm builds on the ideas of mirror descent methods, first introduced by Nemirovski and Yudin [12], which consider updates of the gradient in the dual space. The mirror descent algorithm has been successfully applied in high dimensional problems both in deterministic and stochastic settings [2, 7]. In the present work, we describe a particular instance of the algorithm with an entropy-like proxy function. This method presents similarities with the exponentiated gradient descent algorithm which was derived under different motivations in [10]. A crucial distinction between the two is the additional averaging step in our version which guarantees statistical performance. The idea of averaging recursive procedures is well-known (see e.g. [13] and the references therein) and it has been invoked recently by Zhang [19] for the standard stochastic gradient descent (taking place in the initial parameter space). Also it is worth noticing that most of the existing online methods are evaluated in terms of relative loss bounds which are related to the empirical risk while we focus on generalization error bounds (see [4, 5, 10] for insights on connections between the two types of criteria). The rest of the paper is organized as follows. We first introduce the setup (Section 2), then we describe the algorithm and state the main convergence result (Section 3). Further we provide the intuition underlying the proposed algorithm, and compare it to other methods (Section 4). We end up with a technical section dedicated to the proof of our main result (Section 5).

## 2   Setup and notations

Let $Z$ be a random variable with values in a measurable space $(\mathcal{Z}, \mathcal{A})$. We set a parameter $\lambda > 0$, and an integer $M \geq 2$. The unknown parameter is a vector $\theta \in \mathbb{R}^M$ which is compelled to stay in the decision set $\Theta = \Theta_{M,\lambda}$ defined by:

$$\Theta_{M,\lambda} = \left\{ \theta = (\theta^{(1)}, \dots, \theta^{(M)})^T \in \mathbb{R}_+^M \ : \ \sum_{i=1}^M \theta^{(i)} = \lambda \right\} . \tag{1}$$

Now we introduce the loss function $Q : \Theta \times \mathcal{Z} \to \mathbb{R}_+$ such that the random function $Q(\cdot, Z) : \Theta \to \mathbb{R}_+$ is convex for almost all $Z$ and define the convex risk function $A : \Theta \to \mathbb{R}_+$ to be minimized as follows:

$$A(\theta) = \mathbb{E}\, Q(\theta, Z) . \tag{2}$$

Assume a training sample is given in the form of a sequence $(Z_1, \dots, Z_{t-1})$, where each $Z_i$ has the same distribution as $Z$. We assume for simplicity that the training sequence is i.i.d. though this assumption can be weakened.

We propose to minimize the convex target function $A$ over the decision set $\Theta$ on the basis of the stochastic sub-gradients of $Q$:

$$u_i(\theta) = \nabla_\theta Q(\theta, Z_i), \quad i = 1, 2, \dots, \tag{3}$$

Note that the expectations $\mathbb{E}\, u_i(\cdot)$ belong to the sub-differential of $A(\cdot)$.

In the sequel, we will characterize the accuracy of an estimate $\widehat{\theta}_t = \widehat{\theta}_t(Z_1, \dots, Z_{t-1}) \in \Theta$ of the minimizer of $A$ by the excess risk:

$$\mathbb{E}\, A(\widehat{\theta}_t) - \min_{\theta \in \Theta} A(\theta) \tag{4}$$

where the expectation is taken over the sample $(Z_1, \dots, Z_{t-1})$.

We now introduce the notation that is necessary to present the algorithm in the next section.

For a vector $z = \left(z^{(1)}, \ldots, z^{(M)}\right)^T \in \mathbb{R}^M$, define the norms

$$\|z\|_1 \stackrel{\text{def}}{=} \sum_{j=1}^M |z^{(j)}|, \quad \|z\|_\infty \stackrel{\text{def}}{=} \max_{\|\theta\|_1=1} z^T\theta = \max_{j=1,\ldots,M} |z^{(j)}|.$$

The space $\mathbb{R}^M$ equipped with the norm $\|\cdot\|_1$ is called the primal space $E$ and the same space equipped with the dual norm $\|\cdot\|_\infty$ is called the dual space $E^*$.

Introduce a so-called entropic proxy function:

$$\forall\,\theta \in \Theta, \quad V(\theta) = \lambda \ln\left(M/\lambda\right) + \sum_{j=1}^M \theta^{(j)} \ln \theta^{(j)}, \tag{5}$$

which has its minimum at $\theta_0 = (\lambda/M, \ldots, \lambda/M)^T$. It is easy to check that this function is $\alpha$-strongly convex with respect to the norm $\|\cdot\|_1$ with parameter $\alpha = 1/\lambda$, i.e.,

$$V(sx + (1-s)y) \leq sV(x) + (1-s)V(y) - \frac{\alpha}{2}s(1-s)\|x-y\|_1^2 \tag{6}$$

for all $x, y \in \Theta$ and any $s \in [0,1]$.

Let $\beta > 0$ be a parameter. We call $\beta$-*conjugate* of $V$ the following convex transform:

$$\forall\,z \in \mathbb{R}^M, \quad W_\beta(z) \stackrel{\text{def}}{=} \sup_{\theta\in\Theta} \left\{-z^T\theta - \beta V(\theta)\right\}.$$

As it straightforwardly follows from (5), the $\beta$-conjugate is given here by:

$$W_\beta(z) = \lambda\beta \ln\left(\frac{1}{M}\sum_{k=1}^M e^{-z^{(k)}/\beta}\right), \quad \forall\,z \in \mathbb{R}^M, \tag{7}$$

which has a Lipschitz-continuous gradient w.r.t. $\|\cdot\|_1$, namely,

$$\|\nabla W_\beta(z) - \nabla W_\beta(\tilde{z})\|_1 \leq \frac{\lambda}{\beta}\|z - \tilde{z}\|_\infty, \quad \forall\,z, \tilde{z} \in \mathbb{R}^M. \tag{8}$$

Though we will focus on a particular algorithm based on the entropic proxy function, our results apply for a generic algorithmic scheme which takes advantage of the general properties of convex transforms (see [8] for details). The key property in the proof is the inequality (8).

## 3   Algorithm and main result

The mirror descent algorithm is a stochastic gradient algorithm in the dual space. At each iteration $i$, a new data point $(X_i, Y_i)$ is observed and there are two updates: one is the value $\zeta_i$ as the result of the stochastic gradient descent in the dual space, the other is the update of the parameter $\theta_i$ which is the "mirror image" of $\zeta_i$. In order to tune the algorithm properly, we need two fixed positive sequences $(\gamma_i)_{i\geq1}$ (stepsize) and $(\beta_i)_{i\geq1}$ (temperature) such that $\beta_i \geq \beta_{i-1}$. The *mirror descent algorithm with averaging* is as follows:

**Algorithm.**

- *Fix the initial values $\theta_0 \in \Theta$ and $\zeta_0 = 0 \in \mathbb{R}^M$.*
- *For $i = 1, \ldots, t-1$, do*

$$\begin{aligned}
\zeta_i &= \zeta_{i-1} + \gamma_i u_i(\theta_{i-1}), \\
\theta_i &= -\nabla W_{\beta_i}(\zeta_i).
\end{aligned} \tag{9}$$

- *Output at iteration $t$ the following convex combination:*

$$\hat{\theta}_t = \sum_{i=1}^{t} \gamma_i \theta_{i-1} \Big/ \sum_{j=1}^{t} \gamma_j \, . \tag{10}$$

At this point, we actually have described a class of algorithms. Given the observations of the stochastic sub-gradient (3), particular choices of the proxy function $V$, of the stepsize and temperature parameters, will determine the algorithm completely. We discuss these choices with more details in [8]. In this paper, we focus on the entropic proxy function and consider a nearly optimal choice for the stepsize and temperature parameters which is the following:

$$\gamma_i \equiv 1 \, , \quad \beta_i = \beta_0 \sqrt{i+1} \, , \quad i = 1, 2, \dots , \quad \beta_0 > 0 \, . \tag{11}$$

We can now state our rate of convergence result.

**Theorem.** *Assume that the loss function $Q$ satisfies the following boundedness condition:*

$$\sup_{\theta \in \Theta} \mathbb{E} \, \|\nabla_\theta Q(\theta, Z)\|_\infty^2 \leq L^2 < \infty \, . \tag{12}$$

*Fix also $\beta_0 = L/\sqrt{\ln M}$.*

*Then, for any integer $t \geq 1$, the excess risk of the estimate $\hat{\theta}_t$ described above satisfies the following bound:*

$$\mathbb{E} \, A(\hat{\theta}_t) - \min_{\theta \in \Theta} A(\theta) \leq 2 \, L\lambda \, (\ln M)^{1/2} \, \frac{\sqrt{t+1}}{t} \, . \tag{13}$$

**Example.** Consider the setting of supervised learning where the data are modelled by a pair $(X, Y)$ with $X \in \mathcal{X}$ being an observation vector and $Y$ a label, either integer (classification) or real-valued (regression). Boosting and SVM algorithms are related to the minimization of a functional

$$R(f) = \mathbb{E}\varphi(Yf(X))$$

where $\varphi$ is a convex non-negative cost function (typically exponential, logit or hinge loss) and $f$ belongs to a given class of combined predictors. The aggregation problem consists in finding the best linear combination of elements from a finite set of predictors $\{h_1, \dots, h_M\}$ with $h_j : \mathcal{X} \to [-K, K]$. Taking compact notations, it means that we search for $f$ of the form $f = \theta^T H$ with $H$ denoting the vector-valued function whose components are these base predictors:

$$H(x) = (h_1(x), \dots, h_M(x))^T \, ,$$

and $\theta$ belonging in a decision set $\Theta = \Theta_{M,\lambda}$. Take for instance $\varphi$ to be non-increasing. It is easy to see that this problem can be interpreted in terms of our general setting with $Z = (X, Y)$, $Q(Z, \theta) = \varphi(Y\theta^T H(X))$ and $L = K\varphi'(K\lambda)$. ∎

## 4  Discussion

In this section, we provide some insights on the method and the result of the previous section.

### 4.1  Heuristics

Suppose that we want to minimize a convex function $\theta \mapsto A(\theta)$ over a convex set $\Theta$. If $\theta_0, \dots, \theta_{t-1}$ are the available search points at iteration $t$, we can provide the affine approximations $\phi_i$ of the function $A$ defined, for $\theta \in \Theta$, by

$$\phi_i(\theta) = A(\theta_{i-1}) + (\theta - \theta_{i-1})^T \nabla A(\theta_{i-1}), \quad i = 1, \dots, t \, .$$

Here $\theta \mapsto \nabla A(\theta)$ is a vector function belonging to the sub-gradient of $A(\cdot)$. Taking a convex combination of the $\phi_i$'s, we obtain an averaged approximation of $A(\theta)$:

$$\bar{\phi}_t(\theta) = \frac{\sum_{i=1}^t \gamma_i \left( A(\theta_{i-1}) + (\theta - \theta_{i-1})^T \nabla A(\theta_{i-1}) \right)}{\sum_{i=1}^t \gamma_i}.$$

At first glance, it would seem reasonable to choose as the next search point a vector $\theta \in \Theta$ minimizing the approximation $\bar{\phi}_t$, i.e.,

$$\theta_t = \arg\min_{\theta \in \Theta} \bar{\phi}_t(\theta) = \arg\min_{\theta \in \Theta} \theta^T \left( \sum_{i=1}^t \gamma_i \nabla A(\theta_{i-1}) \right). \tag{14}$$

However, this does not make any progress, because our approximation is "good" only in the vicinity of search points $\theta_0, \ldots, \theta_{t-1}$. Therefore, it is necessary to modify the criterion, for instance, by adding a special penalty $B_t(\theta, \theta_{t-1})$ to the target function in order to keep the next search point $\theta_t$ in the desired region. Thus, one chooses the point:

$$\theta_t = \arg\min_{\theta \in \Theta} \left[ \theta^T \left( \sum_{i=1}^t \gamma_i \nabla A(\theta_{i-1}) \right) + B_t(\theta, \theta_{t-1}) \right]. \tag{15}$$

Our algorithm corresponds to a specific type of penalty $B_t(\theta, \theta_{t-1}) = \beta_t V(\theta)$, where $V$ is the proxy function. Also note that in our problem the vector-function $\nabla A(\cdot)$ is not available. Therefore, we replace in (15) the unknown gradients $\nabla A(\theta_{i-1})$ by the observed stochastic sub-gradients $u_i(\theta_{i-1})$. This yields a new definition of the $t$-th search point:

$$\theta_t = \arg\min_{\theta \in \Theta} \left[ \theta^T \left( \sum_{i=1}^t \gamma_i u_i(\theta_{i-1}) \right) + \beta_t V(\theta) \right] = \arg\max_{\theta \in \Theta} \left[ -\zeta_t^T \theta - \beta_t V(\theta) \right], \tag{16}$$

where $\zeta_t = \sum_{i=1}^t \gamma_i u_i(\theta_{i-1})$. By a standard result of convex analysis (see e.g. [3]), the solution to this problem reads as $-\nabla W_{\beta_t}(\zeta_t)$ and it is now easy to deduce the iterative scheme (9) of the mirror descent algorithm.

## 4.2  Comparison with previous work

The versions of mirror descent method proposed in [12] are somewhat different from our iterative scheme (9). One of them, closest to ours, is studied in detail in [3]. It is based on the recursive relation

$$\theta_i = -\nabla W_1 \left( -\nabla V(\theta_{i-1}) + \gamma_i u_i(\theta_{i-1}) \right), \quad i = 1, 2, \ldots, \tag{17}$$

where the function $V$ is strongly convex with respect to the norm of initial space $E$ (which is not necessarily the space $\ell_1^M$) and $W_1$ is the 1-conjugate function to $V$.

If $\Theta = \mathbb{R}^M$ and $V(\theta) = \frac{1}{2}\|\theta\|_2^2$, the scheme of (17) coincides with the ordinary gradient method.

For the unit simplex $\Theta = \Theta_{M,1}$ and the entropy type proxy function $V$ from (5) with $\lambda = 1$, the coordinates $\theta_i^{(j)}$ of vector $\theta_i$ from (17) are:

$$\forall j = 1, \ldots, M, \quad \theta_i^{(j)} = \frac{\theta_0^{(j)} \exp \left( -\sum_{m=1}^i \gamma_m u_{m,j}(\theta_{m-1}) \right)}{\sum_{k=1}^M \theta_0^{(k)} \exp \left( -\sum_{m=1}^i \gamma_m u_{m,k}(\theta_{m-1}) \right)}. \tag{18}$$

The algorithm is also known as the exponentiated gradient (EG) method [10]. The differences between the algorithm (17) and ours are the following:

- the initial iterative scheme of the Algorithm is different than that of (17), particularly, it includes the second tuning parameter $\beta_i$; moreover, the algorithm (18) uses initial value $\theta_0$ in a different manner;
- our algorithm contains the additional averaging step of the updates (10).

The convergence properties of the EG method (18) have been studied in a deterministic setting [6]. Namely, it has been shown that, under some assumptions, the difference $A_t(\theta_t) - \min_{\theta \in \Theta_{M,1}} A_t(\theta)$, where $A_t$ is the empirical risk, is bounded by a constant depending on $M$ and $t$. If this constant is small enough, these results show that the EG method provides good numerical minimizers of the empirical risk $A_t$. The averaging step allows the use of the results provided in [5] to derive generalization error bounds from relative loss bounds. This technique leads to rates of convergence of the order $\sqrt{(\ln M)/t}$ as well but with suboptimal multiplicative factor in $\lambda$.

Finally, we point out that the algorithm (17) may be deduced from the ideas mentioned in Subsection 4.1 and which are studied in the literature on proximal methods within the field of convex optimization (see, e.g., [9, 1] and the references therein). Namely, under rather general conditions, the variable $\theta_i$ from (17) solves the the minimization problem

$$\theta_i = \underset{\theta \in \Theta}{\arg\min} \left( \theta^T \gamma_i u_i(\theta_{i-1}) + B(\theta, \theta_{i-1}) \right), \tag{19}$$

where the penalty $B(\theta, \theta_{i-1}) = V(\theta) - V(\theta_{i-1}) - (\theta - \theta_{i-1})^T \nabla V(\theta_{i-1})$ represents the Bregman divergence between $\theta$ and $\theta_{i-1}$ related to the function $V$.

### 4.3  General comments

**Performance and efficiency.** The rate of convergence of order $\sqrt{\ln M}/\sqrt{t}$ is typical without low noise assumptions (as they are introduced in [17]). Batch procedures based on minimization of the empirical convex risk functional present a similar rate. From the statistical point of view, there is no remarkable difference between batch and our mirror-descent procedure. On the other hand, from the computational point of view, our procedure is quite comparable with the direct stochastic gradient descent. However, the mirror-descent algorithm presents two major advantages as compared both to batch and to direct stochastic gradient: (i) its behavior with respect to the cardinality of the base class is better than for direct stochastic gradient descent (of the order of $\sqrt{\ln M}$ in the Theorem, instead of $M$ or $\sqrt{M}$ for direct stochastic gradient); (ii) mirror-descent presents a higher efficiency especially in high-dimensional problems as its algorithmic complexity and memory requirements are of strictly smaller order than for corresponding batch procedures (see [7] for a comparison).

**Optimality of the rate of convergence.** Using the techniques of [7] and [16] it is not hard to prove minimax lower bound on the excess risk $\mathbb{E}\, A(\widehat{\theta}_t) - \min_{\theta \in \Theta_{M,\lambda}} A(\theta)$ having the order $(\ln M)^{1/2}/\sqrt{t}$ for $M \geq t^{1/2+\delta}$ with some $\delta > 0$. This indicates that the upper bound of the Theorem is rate optimal for such values of $M$.

**Choice of the base class.** We point out that the good behaviour of this method crucially relies on the choice of the base class of functions $\{h_j\}_{1 \leq j \leq M}$. As far as theory is concerned, in order to provide a complete statistical analysis, one should establish approximation error bounds on the quantity $\inf_{f \in \mathcal{F}_{M,\lambda}} A(f) - \inf_f A(f)$ showing that the richness of the base class is reflected both by diversity (orthogonality or independence) of the $h_j$'s and by its cardinality $M$. For example, one can take $h_j$'s as the eigenfunctions associated to some positive definite kernel. We refer to [14], [15], for related results. The choice of $\lambda$ can be motivated by similar considerations. In fact, to minimize the approximation error it might be useful to take $\lambda$ depending on the sample size $t$ and tending to infinity with some slow rate as in [11]. A balance between the stochastic error as given in the Theorem and the approximation error would then determine the optimal choice of $\lambda$.

# 5 Proof of the Theorem

Introduce the notation $\nabla A(\theta) = \mathbb{E}u_i(\theta)$ and $\xi_i(\theta) = u_i(\theta) - \nabla A(\theta)$. Put $v_i = u_i(\theta_{i-1})$ which gives $\zeta_i - \zeta_{i-1} = \gamma_i v_i$. By continuous differentiability of $W_{\beta_{t-1}}$ and by (8) we have:

$$
\begin{aligned}
W_{\beta_{i-1}}(\zeta_i) &= W_{\beta_{i-1}}(\zeta_{i-1}) + \gamma_i v_i^T \nabla W_{\beta_{i-1}}(\zeta_{i-1}) \\
&\quad + \gamma_i \int_0^1 v_i^T \left[ \nabla W_{\beta_{i-1}}(\tau\zeta_i + (1-\tau)\zeta_{i-1}) - \nabla W_{\beta_{i-1}}(\zeta_{i-1}) \right] \mathrm{d}\tau \\
&\leq W_{\beta_{i-1}}(\zeta_{i-1}) + \gamma_i v_i^T \nabla W_{\beta_{i-1}}(\zeta_{i-1}) + \frac{\lambda\gamma_i^2 \|v_i\|_\infty^2}{2\beta_{i-1}} \ .
\end{aligned}
$$

Then, using the fact that $(\beta_i)_{i\geq 1}$ is a non-decreasing sequence and that, for $z$ fixed, $\beta \mapsto W_\beta(z)$ is a non-increasing function, we get

$$
W_{\beta_i}(\zeta_i) \leq W_{\beta_{i-1}}(\zeta_i) \leq W_{\beta_{i-1}}(\zeta_{i-1}) - \gamma_i \theta_{i-1}^T v_i + \frac{\lambda\gamma_i^2 \|v_i\|_\infty^2}{2\beta_{i-1}} \ .
$$

Summing up over the $i$'s and using the representation $\zeta_t = \sum_{i=1}^t \gamma_i v_i$, we get:

$$
\forall \theta \in \Theta, \quad \sum_{i=1}^t \gamma_i(\theta_{i-1} - \theta)^T v_i \leq -W_{\beta_t}(\zeta_t) - \zeta_t^T \theta + \sum_{i=1}^t \frac{\lambda\gamma_i^2 \|v_i\|_\infty^2}{2\beta_{i-1}}
$$

since $W_{\beta_0}(\zeta_0) = 0$. From definition of $W_\beta$, we have, $\forall \zeta \in \mathbb{R}^M$ and $\forall \theta \in \Theta$, $-W_{\beta_t}(\zeta) - \zeta^T \theta \leq \beta_t V(\theta)$. Finally, since $v_i = \nabla A(\theta_{i-1}) + \xi_i(\theta_{i-1})$, we get

$$
\sum_{i=1}^t \gamma_i(\theta_{i-1} - \theta)^T \nabla A(\theta_{i-1}) \leq \beta_t V(\theta) - \sum_{i=1}^t \gamma_i(\theta_{i-1} - \theta)^T \xi_i(\theta_{i-1}) + \sum_{i=1}^t \frac{\lambda\gamma_i^2 \|v_i\|_\infty^2}{2\beta_{i-1}} \ .
$$

As we are to take expectations, we note that, conditioning on $\theta_{i-1}$ and using the independence between $\theta_{i-1}$ and $(X_i, Y_i)$, we have: $\mathbb{E}\left( (\theta_{i-1} - \theta)^T \xi_i(\theta_{i-1}) \right) = 0$. Now, convexity of $A$ and the previous display lead to:

$$
\begin{aligned}
\forall \theta \in \Theta, \quad \mathbb{E} A(\widehat{\theta}_t) - A(\theta) &\leq \frac{\sum_{i=1}^t \gamma_i \mathbb{E}\left[ (\theta_{i-1} - \theta)^T \nabla A(\theta_{i-1}) \right]}{\sum_{i=1}^t \gamma_i} \\
&= \frac{1}{t} \sum_{i=1}^t \mathbb{E}\left[ (\theta_{i-1} - \theta)^T \nabla A(\theta_{i-1}) \right] \\
&\leq \frac{\sqrt{t+1}}{t} \left( \beta_0 V^* + \frac{\lambda L^2}{\beta_0} \right) \ ,
\end{aligned}
$$

where we have set $V^* = \max_{\theta \in \Theta} V(\theta)$ and made use of the boundedness assumption $\mathbb{E}\|u_i(\theta)\|_\infty^2 \leq L^2$ and of the particular choice for the stepsize and temperature parameters. Noticing that $V^* = \lambda \ln M$ and optimizing this bound in $\beta_0 > 0$, we obtain the result.

**Acknowledgments**

We thank Nicolò Cesa-Bianchi for sharing with us his expertise on relative loss bounds.

# References

[1] Beck, A. & Teboulle, M. (2003) Mirror descent and nonlinear projected subgradient methods for convex optimization. *Operations Research Letters*, 31:167–175.

[2] Ben-Tal, A., Margalit, T. & Nemirovski, A. (2001) The Ordered Subsets Mirror Descent optimization method and its use for the Positron Emission Tomography reconstruction problem. *SIAM J. on Optimization*, 12:79–108.

[3] Ben-Tal, A. & Nemirovski, A.S. (1999) The conjugate barrier mirror descent method for non-smooth convex optimization. MINERVA Optimization Center Report, Technion Institute of Technology.
Available at http://iew3.technion.ac.il/Labs/Opt/opt/Pap/CP_MD.pdf

[4] Cesa-Bianchi, N. & Gentile, C. (2005) Improved risk tail bounds for on-line algorithms. Submitted.

[5] Cesa-Bianchi, N., Conconi, A. & Gentile, C. (2004) On the generalization ability of on-line learning algorithms. *IEEE Transactions on Information Theory*, 50(9):2050–2057.

[6] Helmbold, D.P., Kivinen, J. & Warmuth, M.K. (1999) Relative loss bounds for single neurons. *IEEE Trans. on Neural Networks*, 10(6):1291–1304.

[7] Juditsky, A. & Nemirovski, A. (2000) Functional aggregation for nonparametric estimation. Annals of Statistics, 28(3): 681–712.

[8] Juditsky, A.B., Nazin, A.V., Tsybakov, A.B. & Vayatis N. (2005) Recursive Aggregation of Estimators via the Mirror Descent Algorithm with Averaging. Technical Report LPMA, Université Paris 6.
Available at http://www.proba.jussieu.fr/pageperso/vayatis/publication.html

[9] Kiwiel, K.C. (1997) Proximal minimization methods with generalized Bregman functions. *SIAM J. Control Optim.*, 35:1142–1168.

[10] Kivinen J. & Warmuth M.K. (1997) Additive versus exponentiated gradient updates for linear prediction. *Information and Computation*, Vol.132(1): 1–64.

[11] Lugosi, G. & Vayatis, N. (2004) On the Bayes-risk consistency of regularized boosting methods (with discussion). *Annals of Statitics*, 32(1): 30–55.

[12] Nemirovski, A.S. & Yudin, D.B. (1983) *Problem Complexity and Method Efficiency in Optimization*. Wiley-Interscience.

[13] Polyak, B.T. & Juditsky, A.B. (1992) Acceleration of stochastic approximation by averaging. *SIAM J. Control Optim.*, 30:838–855.

[14] Scovel, J.C. & Steinwart, I. (2005) Fast Rates for Support Vector Machines. In Proceedings of the 18th Conference on Learning Theory (COLT 2005), Bertinoro, Italy.

[15] Tarigan, B. & van de Geer, S. (2004) Adaptivity of Support Vector Machines with $\ell_1$ Penalty. Preprint, University of Leiden.

[16] Tsybakov, A. (2003) Optimal Rates of Aggregation. Proceedings of COLT'03, LNCS, Springer, Vol. 2777:303–313.

[17] Tsybakov, A. (2004) Optimal aggregation of classifiers in statistical learning. *Annals of Statistics*, 32(1):135–166.

[18] Zhang, T. (2004) Statistical behavior and consistency of classification methods based on convex risk minimization (with discussion). *Annals of Statistics*, 32(1):56–85.

[19] Zhang, T. (2004) Solving large scale linear prediction problems using stochastic gradient descent algorithms. In Proceedings of ICML'04.
